# Forward dynamic models in human motor control: Psychophysical evidence

**Daniel M. Wolpert**
wolpert@psyche.mit.edu

**Zoubin Ghahramani**
zoubin@psyche.mit.edu

**Michael I. Jordan**
jordan@psyche.mit.edu

Department of Brain & Cognitive Sciences
Massachusetts Institute of Technology
Cambridge, MA 02139

## Abstract

Based on computational principles, with as yet no direct experimental validation, it has been proposed that the central nervous system (CNS) uses an internal model to simulate the dynamic behavior of the motor system in planning, control and learning (Sutton and Barto, 1981; Ito, 1984; Kawato et al., 1987; Jordan and Rumelhart, 1992; Miall et al., 1993). We present experimental results and simulations based on a novel approach that investigates the temporal propagation of errors in the sensorimotor integration process. Our results provide direct support for the existence of an internal model.

## 1   Introduction

The notion of an internal model, a system which mimics the behavior of a natural process, has emerged as an important theoretical concept in motor control (Jordan, 1995). There are two varieties of internal models—"forward models," which mimic the causal flow of a process by predicting its next state given the current state and the motor command, and "inverse models," which are anticausal, estimating the motor command that causes a particular state transition. Forward models— the focus of this article—have been been shown to be of potential use for solving four fundamental problems in computational motor control. First, the delays in most sensorimotor loops are large, making feedback control infeasible for rapid

movements. By using a forward model for internal feedback the outcome of an action can be estimated and used before sensory feedback is available (Ito, 1984; Miall et al., 1993). Second, a forward model is a key ingredient in a system that uses motor outflow ("efference copy") to anticipate and cancel the reafferent sensory effects of self-movement (Gallistel, 1980; Robinson et al., 1986). Third, a forward model can be used to transform errors between the desired and actual sensory outcome of a movement into the corresponding errors in the motor command, thereby providing appropriate signals for motor learning (Jordan and Rumelhart, 1992). Similarly by predicting the sensory outcome of the action, without actually performing it, a forward model can be used in mental practice to learn to select between possible actions (Sutton and Barto, 1981). Finally, a forward model can be used for state estimation in which the model's prediction of the next state is combined with a reafferent sensory correction (Goodwin and Sin, 1984). Although shown to be of theoretical importance, the existence and use of an internal forward model in the CNS is still a major topic of debate.

When a subject moves his arm in the dark, he is able to estimate the visual location of his hand with some degree of accuracy. Observer models from engineering formalize the sources of information which the CNS could use to construct this estimate (Figure 1). This framework consists of a state estimation process (the observer) which is able to monitor both the inputs and outputs of the system. In particular, for the arm, the inputs are motor commands and the output is sensory feedback (e.g. vision and proprioception). There are three basic methods whereby the observer can estimate the current state (e.g. position and velocity) of the hand form these sources: It can make use of sensory inflow, it can make use of integrated motor outflow (dead reckoning), or it can combine these two sources of information via the use of a forward model.

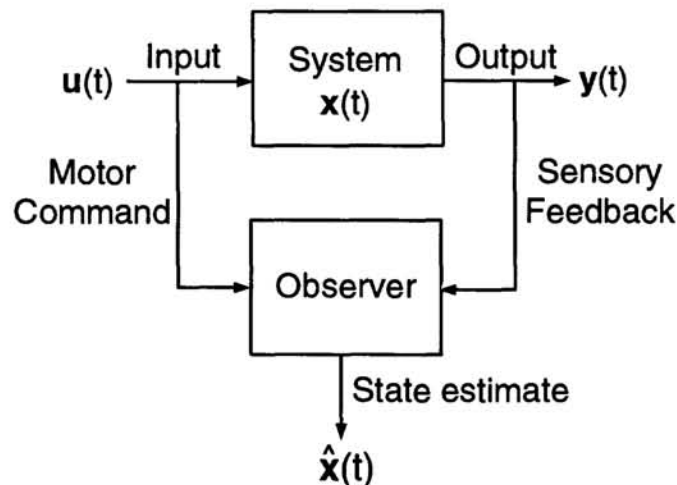

Figure 1. Observer model of state estimation.

## 2 State Estimation Experiment

To test between these possibilities, we carried out an experiment in which subjects made arm movements in the dark. The full details of the experiment are described in the Appendix. Three experimental conditions were studied, involving the use of null, assistive and resistive force fields. The subjects' internal estimate of hand location was assessed by asking them to localize visually the position of their hand at the end of the movement. The bias of this location estimate, plotted as a function of movement duration shows a consistent overestimation of the distance moved (Figure 2). This bias shows two distinct phases as a function of movement duration, an initial increase reaching a peak of 0.9 cm after one second followed by a sharp transition to a region of gradual decline. The variance of the estimate also shows an initial increase during the first second of movement after which it plateaus at about 2 cm$^2$. External forces had distinct effects on the bias and variance propagation. Whereas the bias was increased by the assistive force and decreased by the resistive force ($p < 0.05$), the variance was unaffected.

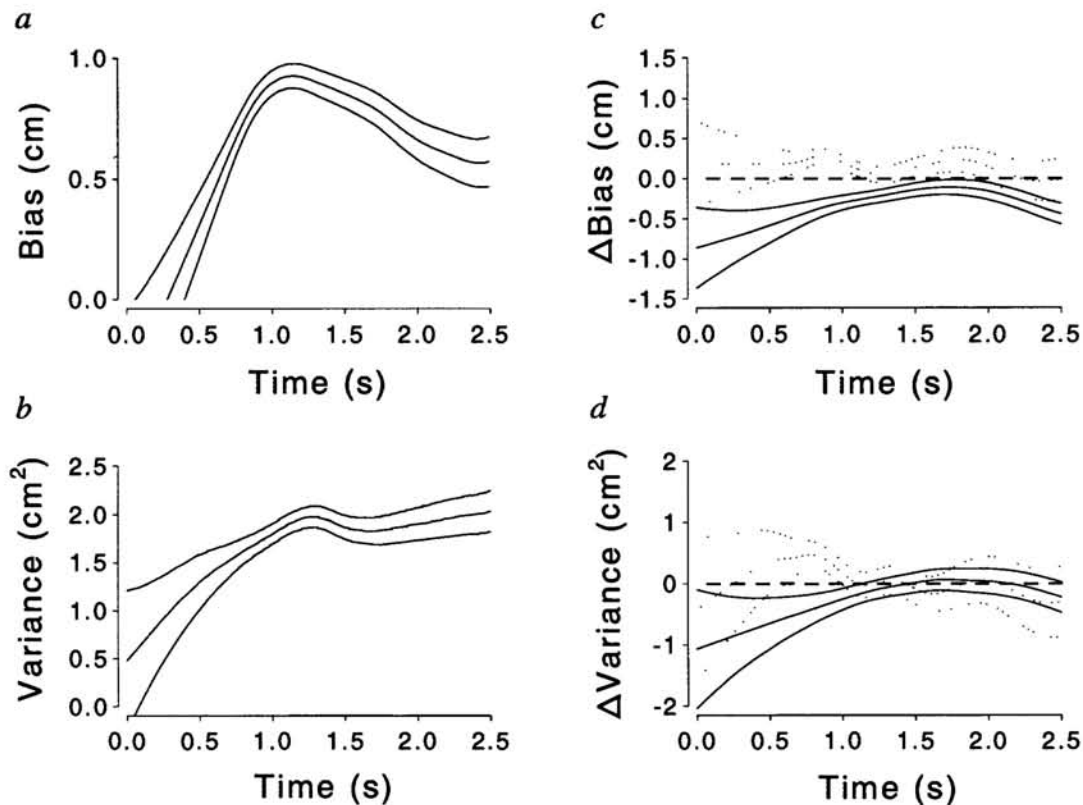

Figure 2. The propagation of the (a) bias and (b) variance of the state estimate is shown, with standard error lines, against movement duration. The differential effects on (c) bias and (d) variance of the external force, assistive (dotted lines) and resistive (solid lines), are also shown relative to zero (dashed line). A positive bias represents an overestimation of the distance moved.

## 3   Observer Model Simulation

These experimental results can be fully accounted for only if we assume that the motor control system integrates the efferent outflow and the reafferent sensory inflow. To establish this conclusion we have developed an explicit model of the sensorimotor integration process which contains as special cases all three of the methods referred to above. The model—a Kalman filter (Kalman and Bucy, 1961)—is a linear dynamical system that produces an estimate of the location of the hand by monitoring both the motor outflow and the feedback as sensed, in the absence of vision, solely by proprioception. Based on these sources of information the model estimates the arm's state, integrating sensory and motor signals to reduce the overall uncertainty in its estimate.

Representing the state of the hand at time $t$ as $\mathbf{x}(t)$ (a $2 \times 1$ vector of position and velocity) acted on by a force $\mathbf{u} = [u_{\text{int}}, u_{\text{ext}}]^T$, combining both internal motor commands and external forces, the system dynamic equations can be written in the general form of

$$\dot{\mathbf{x}}(t) = A\mathbf{x}(t) + B\mathbf{u}(t) + \mathbf{w}(t), \tag{1}$$

where $A$ and $B$ are matrices with appropriate dimension. The vector $\mathbf{w}(t)$ represents the process of white noise with an associated covariance matrix given by $Q = E[\mathbf{w}(t)\mathbf{w}(t)^T]$. The system has an observable output $\mathbf{y}(t)$ which is linked to the actual hidden state $\mathbf{x}(t)$ by

$$\mathbf{y}(t) = C\mathbf{x}(t) + \mathbf{v}(t), \tag{2}$$

where $C$ is a matrix with appropriate dimension and the vector $\mathbf{v}(t)$ represents the output white noise which has the associated covariance matrix $R = E[\mathbf{v}(t)\mathbf{v}(t)^T]$. In our paradigm, $\mathbf{y}(t)$ represents the proprioceptive signals (e.g. from muscle spindles and joint receptors).

In particular, for the hand we approximate the system dynamics by a damped point mass moving in one dimension acted on by a force $\mathbf{u}(t)$. Equation 1 becomes

$$\dot{\mathbf{x}}(t) = \begin{bmatrix} 0 & 1 \\ 0 & -\beta/m \end{bmatrix} \mathbf{x}(t) + \frac{1}{m} \begin{bmatrix} 0 & 0 \\ 1 & 1 \end{bmatrix} \mathbf{u}(t) + \mathbf{w}(t) \tag{3}$$

where the hand has mass $m$ and damping coefficient $\beta$. We assume that this system is fully observable and choose $C$ to be the identity matrix. The parameters in the simulation, $\beta = 3.9$ N·s/m, $m = 4$ kg and $u_{\text{int}} = 1.5$ N were chosen based on the mass of the arm and the observed relationship between time and distance traveled. The external force $u_{\text{ext}}$ was set to $-0.3$, $0$ and $0.3$ N for the resistive, null and assistive conditions respectively. To end the movement the sign of the motor command $u_{\text{int}}$ was reversed until the arm was stationary. Noise covariance matrices of $Q = 9.5 \times 10^{-5}I$ and $R = 3.3 \times 10^{-4}I$ were used representing a standard deviation of 1.0 cm for the position output noise and 1.8 cm s$^{-1}$ for the position component of the state noise.

At time $t = 0$ the subject is given full view of his arm and, therefore, starts with an estimate $\hat{\mathbf{x}}(0) = \mathbf{x}(0)$ with zero bias and variance—we assume that vision calibrates the system. At this time the light is extinguished and the subject must rely on the inputs and outputs to estimate the system's state. The Kalman filter, using a

model of the system $\hat{A}$, $\hat{B}$ and $\hat{C}$, provides an optimal linear estimator of the state given by

$$\dot{\mathbf{x}}(t) = \underbrace{\hat{A}\hat{\mathbf{x}}(t) + \hat{B}\mathbf{u}(t)}_{\text{forward model}} + \underbrace{K(t)[\mathbf{y}(t) - \hat{C}\hat{\mathbf{x}}(t)]}_{\text{sensory correction}}$$

where $K(t)$ is the recursively updated gain matrix. The model is, therefore, a combination of two processes which together contribute to the state estimate. The first process uses the current state estimate and motor command to predict the next state by simulating the movement dynamics with a forward model. The second process uses the difference between actual and predicted reafferent sensory feedback to correct the state estimate resulting from the forward model. The relative contributions of the internal simulation and sensory correction processes to the final estimate are modulated by the Kalman gain matrix $K(t)$ so as to provide optimal state estimates. We used this state update equation to model the bias and variance propagation and the effects of the external force.

By making particular choices for the parameters of the Kalman filter, we are able to simulate dead reckoning, sensory inflow-based estimation, and forward model-based sensorimotor integration. Moreover, to accommodate the observation that subjects generally tend to overestimate the distance that their arm has moved, we set the gain that couples force to state estimates to a value that is larger than its veridical value; $\hat{B} = \frac{1}{m} \begin{bmatrix} 0 & 0 \\ 1.4 & 1.6 \end{bmatrix}$ while both $\hat{A}$ and $\hat{C}$ accurately reflected the true system. This is consistent with the independent data that subjects tend to under-reach in pointing tasks suggesting an overestimation of distance traveled (Soechting and Flanders, 1989).

Simulations of the Kalman filter demonstrate the two distinct phases of bias propagation observed (Figure 3). By overestimating the force acting on the arm the forward model overestimates the distance traveled, an integrative process eventually balanced by the sensory correction. The model also captures the differential effects on bias of the externally imposed forces. By overestimating an increased force under the assistive condition, the bias in the forward model accrues more rapidly and is balanced by the sensory feedback at a higher level. The converse applies to the resistive force. In accord with the experimental results the model predicts no change in variance under the two force conditions.

## 4 Discussion

We have shown that the Kalman filter is able to reproduce the propagation of the bias and variance of estimated position of the hand as a function of both movement duration and external forces. The Kalman filter also simulates the interesting and novel empirical result that while the variance asymptotes, the bias peaks after about one second and then gradually declines. This behavior is a consequence of a trade off between the inaccuracies accumulating in the internal simulation of the arm's dynamics and the feedback of actual sensory information. Simple models which do not trade off the contributions of a forward model with sensory feedback, such as those based purely on sensory inflow or on motor outflow, are unable to reproduce the observed pattern of bias and variance propagation. The ability of the Kalman filter to parsimoniously model our data suggests that the processes embodied in the

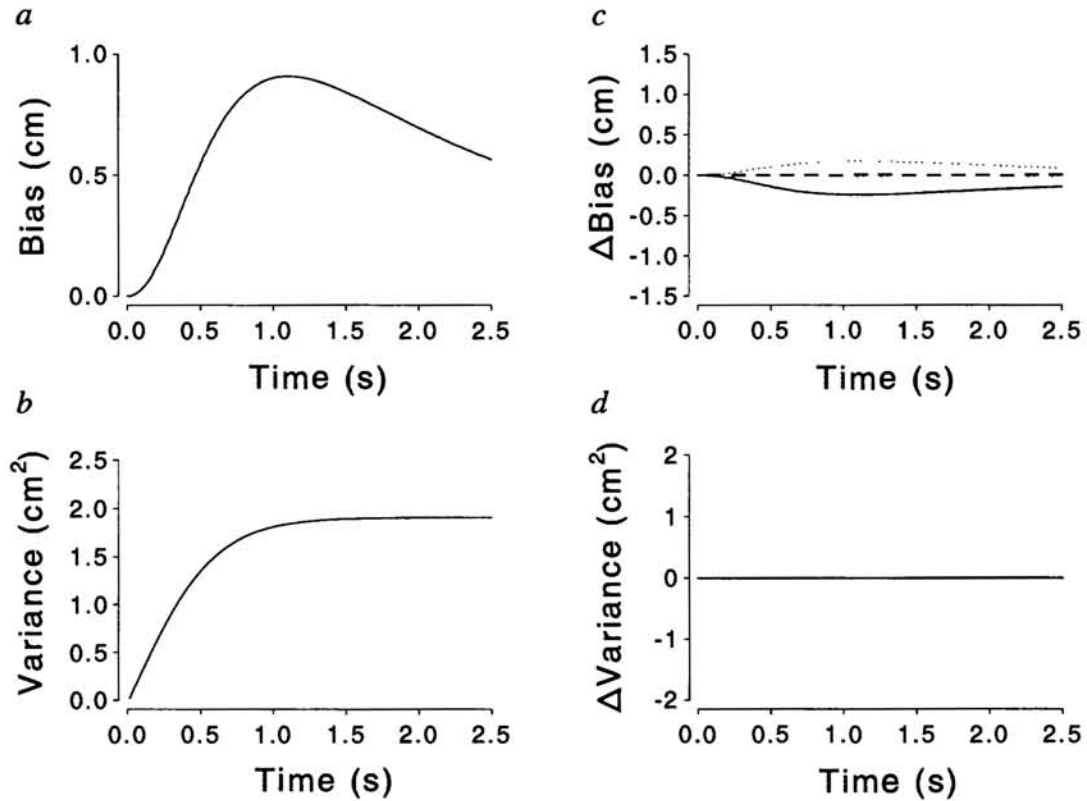

Figure 3.  Simulated bias and variance propagation, in the same representation and scale as Figure 2, from a Kalman filter model of the sensorimotor integration process.

filter, namely internal simulation through a forward model together with sensory correction, are likely to be embodied in the sensorimotor integration process. We feel that the results of this state estimation study provide strong evidence that a forward model is used by the CNS in maintaining its estimate of the hand location. Furthermore, the state estimation paradigm provides a framework to study the sensorimotor integration process in both normal and patient populations. For example, the specific predictions of the sensorimotor integration model can be tested in both patients with sensory neuropathies, who lack proprioceptive reafference, and in patients with damage to the cerebellum, a proposed site for the forward model (Miall et al., 1993).

## Acknowledgements

We thank Peter Dayan for suggestions about the manuscript. This project was supported by grants from the McDonnell-Pew Foundation, ATR Human Information Processing Research Laboratories, Siemens Corporation, and by grant N00014-94-1-0777 from the Office of Naval Research. Daniel M. Wolpert and Zoubin Ghahramani are McDonnell-Pew Fellows in Cognitive Neuroscience. Michael I. Jordan is a NSF Presidential Young Investigator.

## Appendix: Experimental Paradigm

To investigate the way in which errors in the state estimate change over time and with external forces we used a setup (Figure 4) consisting of a combination of planar virtual visual feedback with a two degree of freedom torque motor driven manipulandum (Faye, 1986). The subject held a planar manipulandum on which his thumb was mounted. The manipulandum was used both to accurately measure the position of the subject's thumb and also, using the torque motors, to constrain the hand to move along a line across the subject's body. A projector was used to create virtual images in the plane of the movement by projecting a computer VGA screen onto a horizontal rear projection screen suspended above the manipulandum. A horizontal semi-silvered mirror was placed midway between the screen and manipulandum. The subject viewed the reflected image of the rear projection screen by looking down at the mirror; all projected images, therefore, appeared to be in the plane of the thumb, independent of head position.

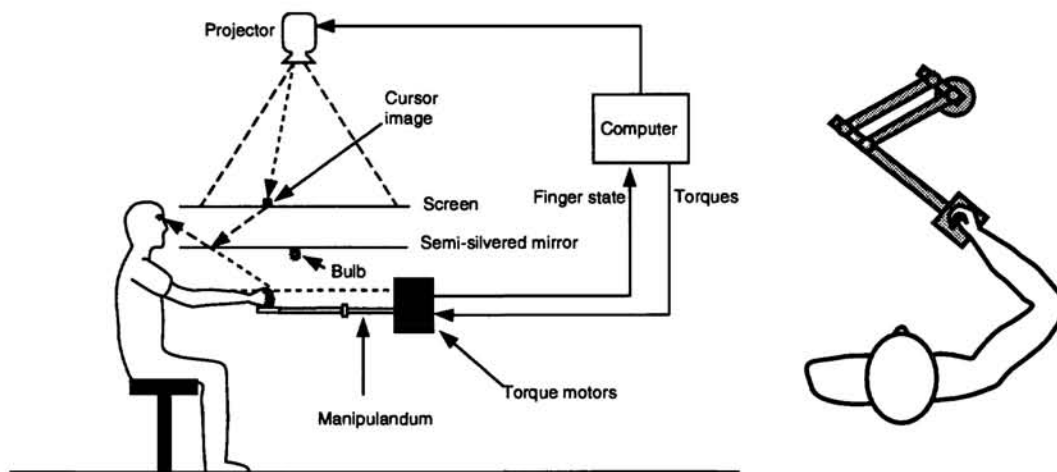

Figure 4. Experimental Setup

Eight subjects participated and performed 300 trials each. Each trial started with the subject visually placing his thumb at a target square projected randomly on the movement line. The arm was then illuminated for two seconds, thereby allowing the subject to perceive visually his initial arm configuration. The light was then extinguished leaving just the initial target. The subject was then required to move his hand either to the left or right, as indicated by an arrow in the initial starting square. This movement was made in the absence of visual feedback of arm configuration. The subject was instructed to move until he heard a tone at which point he stopped. The timing of the tone was controlled to produce a uniform distribution of path lengths from 0–30 cm. During this movement the subject either moved in a randomly selected null or constant assistive or resistive 0.3N force field generated by the torque motors. Although it is not possible to directly probe a subject's internal representation of the state of his arm, we can examine a function of this state—the estimated visual location of the thumb. (The relationship between the state of the arm and the visual coordinates of the hand is known as

the kinematic transformation; Craig, 1986.) Therefore, once at rest the subject indicated the visual estimate of his unseen thumb position using a trackball, held in his other hand, to move a cursor projected in the plane of the thumb along the movement line. The discrepancy between the actual and visual estimate of thumb location was recorded as a measure of the state estimation error. The bias and variance propagation of the state estimate was analyzed as a function of movement duration and external forces. A generalized additive model (Hastie and Tibshirani, 1990) with smoothing splines (five effective degrees of freedom) was fit to the bias and variance as a function of final position, movement duration and the interaction of the two forces with movement duration, simultaneously for main effects and for each subject. This procedure factors out the additive effects specific to each subject and, through the final position factor, the position-dependent inaccuracies in the kinematic transformation.

# References

Craig, J. (1986). *Introduction to robotics*. Addison-Wesley, Reading, MA.

Faye, I. (1986). *An impedence controlled manipulandum for human movement studies*. MS Thesis, MIT Dept. Mechanical Engineering, Cambridge, MA.

Gallistel, C. (1980). *The organization of action: A new synthesis*. Erlbaum, Hilladale, NJ.

Goodwin, G. and Sin, K. (1984). *Adaptive filtering prediction and control*. Prentice-Hall, Englewood Cliffs, NJ.

Hastie, T. and Tibshirani, R. (1990). *Generalized Additive Models*. Chapman and Hall, London.

Ito, M. (1984). *The cerebellum and neural control*. Raven Press, New York.

Jordan, M. and Rumelhart, D. (1992). Forward models: Supervised learning with a distal teacher. *Cognitive Science*, 16:307–354.

Jordan, M. I. (1995). Computational aspects of motor control and motor learning. In Heuer, H. and Keele, S., editors, *Handbook of Perception and Action: Motor Skills*. Academic Press, New York.

Kalman, R. and Bucy, R. S. (1961). New results in linear filtering and prediction. *Journal of Basic Engineering (ASME)*, 83D:95–108.

Kawato, M., Furawaka, K., and Suzuki, R. (1987). A hierarchical neural network model for the control and learning of voluntary movements. *Biol. Cybern.*, 56:1–17.

Miall, R., Weir, D., Wolpert, D., and Stein, J. (1993). Is the cerebellum a Smith Predictor? *Journal of Motor Behavior*, 25(3):203–216.

Robinson, D., Gordon, J., and Gordon, S. (1986). A model of the smooth pursuit eye movement system. *Biol. Cybern.*, 55:43–57.

Soechting, J. and Flanders, M. (1989). Sensorimotor representations for pointing to targets in three- dimensional space. *J. Neurophysiol.*, 62:582–594.

Sutton, R. and Barto, A. (1981). Toward a modern theory of adaptive networks: expectation and prediction. *Psychol. Rev.*, 88:135–170.
